# Bidirectional Recurrent Convolutional Networks for Multi-Frame Super-Resolution

**Yan Huang**[1]     **Wei Wang**[1]     **Liang Wang**[1,2]
[1]Center for Research on Intelligent Perception and Computing
National Laboratory of Pattern Recognition
[2]Center for Excellence in Brain Science and Intelligence Technology
Institute of Automation, Chinese Academy of Sciences
{yhuang, wangwei, wangliang}@nlpr.ia.ac.cn

## Abstract

Super resolving a low-resolution video is usually handled by either single-image super-resolution (SR) or multi-frame SR. Single-Image SR deals with each video frame independently, and ignores intrinsic temporal dependency of video frames which actually plays a very important role in video super-resolution. Multi-Frame SR generally extracts motion information, e.g., optical flow, to model the temporal dependency, which often shows high computational cost. Considering that recurrent neural networks (RNNs) can model long-term contextual information of temporal sequences well, we propose a bidirectional recurrent convolutional network for efficient multi-frame SR. Different from vanilla RNNs, 1) the commonly-used recurrent full connections are replaced with weight-sharing convolutional connections and 2) conditional convolutional connections from previous input layers to the current hidden layer are added for enhancing visual-temporal dependency modelling. With the powerful temporal dependency modelling, our model can super resolve videos with complex motions and achieve state-of-the-art performance. Due to the cheap convolution operations, our model has a low computational complexity and runs orders of magnitude faster than other multi-frame methods.

## 1   Introduction

Since large numbers of high-definition displays have sprung up, generating high-resolution videos from previous low-resolution contents, namely video super-resolution (SR), is under great demand. Recently, various methods have been proposed to handle this problem, which can be classified into two categories: 1) single-image SR [10, 5, 9, 8, 12, 25, 23] super resolves each of the video frames independently, and 2) multi-frame SR [13, 17, 3, 2, 14, 13] models and exploits temporal dependency among video frames, which is usually considered as an essential component of video SR.

Existing multi-frame SR methods generally model the temporal dependency by extracting subpixel motions of video frames, e.g., estimating optical flow based on sparse prior integration or variation regularity [2, 14, 13]. But such accurate motion estimation can only be effective for video sequences which contain small motions. In addition, the high computational cost of these methods limits the real-world applications. Several solutions have been explored to overcome these issues by avoiding the explicit motion estimation [21, 16]. Unfortunately, they still have to perform implicit motion estimation to reduce temporal aliasing and achieve resolution enhancement when large motions are encountered.

Given the fact that recurrent neural networks (RNNs) can well model long-term contextual information for video sequence, we propose a bidirectional recurrent convolutional network (BRCN)

to efficiently learn the temporal dependency for multi-frame SR. The proposed network exploits three convolutions. 1) *Feedforward convolution* models visual spatial dependency between a low-resolution frame and its high-resolution result. 2) *Recurrent convolution* connects the hidden layers of successive frames to learn temporal dependency. Different from the commonly-used full recurrent connection in vanilla RNNs, it is a weight-sharing convolutional connection here. 3) *Conditional convolution* connects input layers at the previous timestep to the current hidden layer, to further enhance visual-temporal dependency modelling. To simultaneously consider the temporal dependency from both previous and future frames, we exploit a forward recurrent network and a backward recurrent network, respectively, and then combine them together for the final prediction. We apply the proposed model to super resolve videos with complex motions. The experimental results demonstrate that the model can achieve state-of-the-art performance, as well as orders of magnitude faster speed than other multi-frame SR methods.

Our main contributions can be summarized as follows. We propose a bidirectional recurrent convolutional network for multi-frame SR, where the temporal dependency can be efficiently modelled by bidirectional recurrent and conditional convolutions. It is an end-to-end framework which does not need pre-/post-processing. We achieve better performance and faster speed than existing multi-frame SR methods.

## 2 Related Work

We will review the related work from the following prospectives.

**Single-Image SR**. Irani and Peleg [10] propose the primary work for this problem, followed by Freeman et al. [8] studying this problem in a learning-based way. To alleviate high computational complexity, Bevilacqua et al. [4] and Chang et al. [5] introduce manifold learning techniques which can reduce the required number of image patch exemplars. For further acceleration, Timofte et al. [23] propose the anchored neighborhood regression method. Yang et al. [25] and Zeyde et al. [26] exploit compressive sensing to encode image patches with a compact dictionary and obtain sparse representations. Dong et al. [6] learn a convolutional neural network for single-image SR which achieves the current state-of-the-art result. In this work, we focus on multi-frame SR by modelling temporal dependency in video sequences.

**Multi-Frame SR**. Baker and Kanade [2] extract optical flow to model the temporal dependency in video sequences for video SR. Then, various improvements [14, 13] around this work are explored to better handle visual motions. However, these methods suffer from the high computational cost due to the motion estimation. To deal with this problem, Protter et al. [16] and Takeda et al. [21] avoid motion estimation by employing nonlocal mean and 3D steering kernel regression. In this work, we propose bidirectional recurrent and conditional convolutions as an alternative to model temporal dependency and achieve faster speed.

## 3 Bidirectional Recurrent Convolutional Network

### 3.1 Formulation

Given a low-resolution, noisy and blurry video, our goal is to obtain a high-resolution, noise-free and blur-free version. In this paper, we propose a bidirectional recurrent convolutional network (BRCN) to map the low-resolution frames to high-resolution ones. As shown in Figure 1, the proposed network contains a forward recurrent convolutional sub-network and a backward recurrent convolutional sub-network to model the temporal dependency from both previous and future frames. Note that similar bidirectional scheme has been proposed previously in [18]. The two sub-networks of BRCN are denoted by two black blocks with dash borders, respectively. In each sub-network, there are four layers including the input layer, the first hidden layer, the second hidden layer and the output layer, which are connected by three convolutional operations:

1. **Feedforward Convolution**. The multi-layer convolutions denoted by black lines learn visual spatial dependency between a low-resolution frame and its high-resolution result. Similar configurations have also been explored previously in [11, 7, 6].

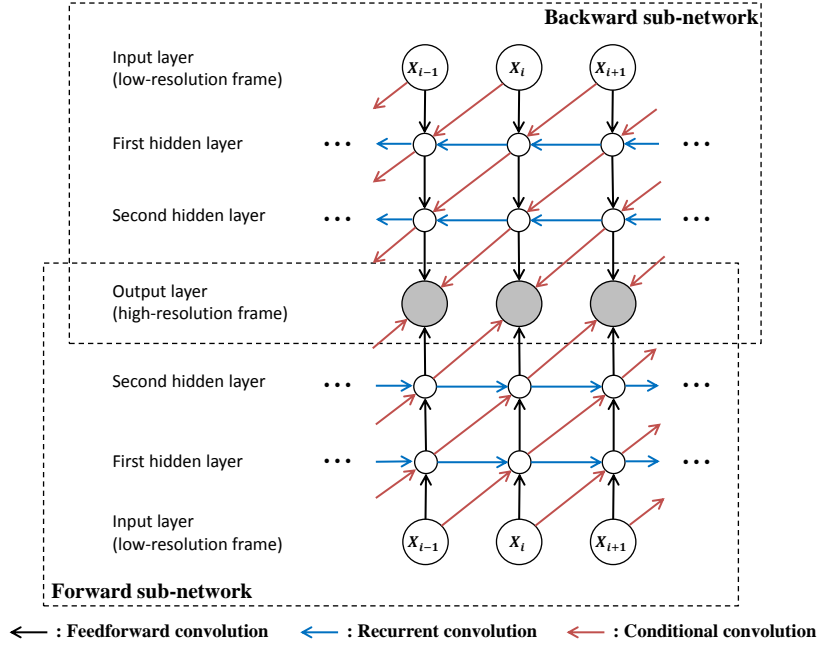

Figure 1: The proposed bidirectional recurrent convolutional network (BRCN).

2. **Recurrent Convolution**. The convolutions denoted by blue lines aim to model long-term temporal dependency across video frames by connecting adjacent hidden layers of successive frames, where the current hidden layer is conditioned on the hidden layer at the previous timestep. We use the recurrent convolution in both forward and backward sub-networks. Such bidirectional recurrent scheme can make full use of the forward and backward temporal dynamics.

3. **Conditional Convolution**. The convolutions denoted by red lines connect input layer at the previous timestep to the current hidden layer, and use previous inputs to provide long-term contextual information. They enhance visual-temporal dependency modelling with this kind of conditional connection.

We denote the frame sets of a low-resolution video[1] $\mathcal{X}$ as $\{\mathbf{X}_i\}_{i=1,2,\ldots,T}$, and infer the other three layers as follows.

**First Hidden Layer**. When inferring the first hidden layer $\mathbf{H}_1^f(\mathbf{X}_i)$ (or $\mathbf{H}_1^b(\mathbf{X}_i)$) at the $i^{th}$ timestep in the forward (or backward) sub-network, three inputs are considered: 1) the current input layer $\mathbf{X}_i$ connected by a feedforward convolution, 2) the hidden layer $\mathbf{H}_1^f(\mathbf{X}_{i-1})$ (or $\mathbf{H}_1^b(\mathbf{X}_{i+1})$) at the $i-1^{th}$ (or $i+1^{th}$) timestep connected by a recurrent convolution, and 3) the input layer $\mathbf{X}_{i-1}$ (or $\mathbf{X}_{i+1}$) at the $i-1^{th}$ (or $i+1^{th}$) timestep connected by a conditional convolution.

$$\mathbf{H}_1^f(\mathbf{X}_i) = \lambda(\mathbf{W}_{v_1}^f * \mathbf{X}_i + \mathbf{W}_{r_1}^f * \mathbf{H}_1^f(\mathbf{X}_{i-1}) + \mathbf{W}_{t_1}^f * \mathbf{X}_{i-1} + \mathbf{B}_1^f)$$
$$\mathbf{H}_1^b(\mathbf{X}_i) = \lambda(\mathbf{W}_{v_1}^b * \mathbf{X}_i + \mathbf{W}_{r_1}^b * \mathbf{H}_1^b(\mathbf{X}_{i+1}) + \mathbf{W}_{t_1}^b * \mathbf{X}_{i+1} + \mathbf{B}_1^b)$$

(1)

where $\mathbf{W}_{v_1}^f$ (or $\mathbf{W}_{v_1}^b$) and $\mathbf{W}_{t_1}^f$ (or $\mathbf{W}_{t_1}^b$) represent the filters of feedforward and conditional convolutions in the forward (or backward) sub-network, respectively. Both of them have the size of $c \times f_{v_1} \times f_{v_1} \times n_1$, where $c$ is the number of input channels, $f_{v_1}$ is the filter size and $n_1$ is the number of filters. $\mathbf{W}_{r_1}^f$ (or $\mathbf{W}_{r_1}^b$) represents the filters of recurrent convolutions. Their filter size $f_{r_1}$ is set to 1 to avoid border effects. $\mathbf{B}_1^f$ (or $\mathbf{B}_1^b$) represents biases. The activation function is the rectified linear unit (ReLu): $\lambda(x) = \max(0, x)$ [15]. Note that in Equation 1, the filter responses of recurrent and

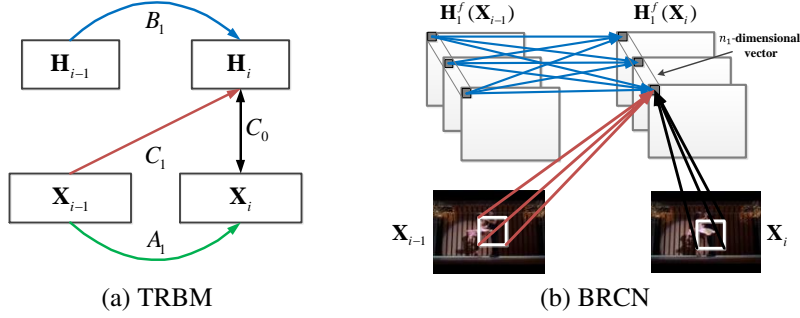

$$(a)\ TRBM \qquad\qquad\qquad (b)\ BRCN$$

Figure 2: Comparison between TRBM and the proposed BRCN.

conditional convolutions can be regarded as dynamic changing biases, which focus on modelling the temporal changes across frames, while the filter responses of feedforward convolution focus on learning visual content.

**Second Hidden Layer**. This phase projects the obtained feature maps $\mathbf{H}_1^f(\mathbf{X}_i)$ (or $\mathbf{H}_1^b(\mathbf{X}_i)$) from $n_1$ to $n_2$ dimensions, which aims to capture the nonlinear structure in sequence data. In addition to intra-frame mapping by feedforward convolution, we also consider two inter-frame mappings using recurrent and conditional convolutions, respectively. The projected $n_2$-dimensional feature maps in the second hidden layer $\mathbf{H}_2^f(\mathbf{X}_i)$ (or $(\mathbf{H}_2^b(\mathbf{X}_i))$ in the forward (or backward) sub-network can be obtained as follows:

$$
\begin{aligned}
\mathbf{H}_2^f(\mathbf{X}_i) &= \lambda(\mathbf{W}_{v_2}^f * \mathbf{H}_1^f(\mathbf{X}_i) + \mathbf{W}_{r_2}^f * \mathbf{H}_2^f(\mathbf{X}_{i-1}) + \mathbf{W}_{t_2}^f * \mathbf{H}_1^f(\mathbf{X}_{i-1}) + \mathbf{B}_2^f) \\
\mathbf{H}_2^b(\mathbf{X}_i) &= \lambda(\mathbf{W}_{v_2}^b * \mathbf{H}_1^b(\mathbf{X}_i) + \mathbf{W}_{r_2}^b * \mathbf{H}_2^b(\mathbf{X}_{i+1}) + \mathbf{W}_{t_2}^b * \mathbf{H}_1^b(\mathbf{X}_{i+1}) + \mathbf{B}_2^b)
\end{aligned}
\tag{2}
$$

where $\mathbf{W}_{v_2}^f$ (or $\mathbf{W}_{v_2}^b$) and $\mathbf{W}_{t_2}^f$ (or $\mathbf{W}_{t_2}^b$) represent the filters of feedforward and conditional convolutions, respectively, both of which have the size of $n_1 \times 1 \times 1 \times n_2$. $\mathbf{W}_{r_2}^f$ (or $\mathbf{W}_{r_2}^b$) represents the filters of recurrent convolution, whose size is $n_2 \times 1 \times 1 \times n_2$.

Note that the inference of the two hidden layers can be regarded as a representation learning phase, where we could stack more hidden layers to increase the representability of our network to better capture the complex data structure.

**Output Layer**. In this phase, we combine the projected $n_2$-dimensional feature maps in both forward and backward sub-networks to jointly predict the desired high-resolution frame:

$$
\mathbf{O}(\mathbf{X}_i) = \mathbf{W}_{v_3}^f * \mathbf{H}_2^f(\mathbf{X}_i) + \mathbf{W}_{t_3}^f * \mathbf{H}_2^f(\mathbf{X}_{i-1}) + \mathbf{B}_3^f + \mathbf{W}_{v_3}^b * \mathbf{H}_2^b(\mathbf{X}_i) + \mathbf{W}_{t_3}^b * \mathbf{H}_2^b(\mathbf{X}_{i+1}) + \mathbf{B}_3^b
\tag{3}
$$

where $\mathbf{W}_{v_3}^f$ (or $\mathbf{W}_{v_3}^b$) and $\mathbf{W}_{t_3}^f$ (or $\mathbf{W}_{t_3}^b$) represent the filters of feedforward and conditional convolutions, respectively. Their sizes are both $n_2 \times f_{v_3} \times f_{v_3} \times c$. We do not use any recurrent convolution for output layer.

## 3.2 Connection with Temporal Restricted Boltzmann Machine

In this section, we discuss the connection between the proposed BRCN and temporal restricted boltzmann machine (TRBM) [20] which is a widely used model in sequence modelling.

As shown in Figure 2, TRBM and BRCN contain similar recurrent connections (blue lines) between hidden layers, and conditional connections (red lines) between input layer and hidden layer. They share the common flexibility to model and propagate temporal dependency along the time. However, TRBM is a generative model while BRCN is a discriminative model, and TRBM contains an additional connection (green line) between input layers for sample generation.

In fact, BRCN can be regarded as a deterministic, bidirectional and patch-based implementation of TRBM. Specifically, when inferring the hidden layer in BRCN, as illustrated in Figure 2 (b), feedforward and conditional convolutions extract overlapped patches from the input, each of which is

fully connected to a $n_1$-dimensional vector in the feature maps $\mathbf{H}_1^f(\mathbf{X}_i)$. For recurrent convolutions, since each filter size is 1 and all the filters contain $n_1 \times n_1$ weights, a $n_1$-dimensional vector in $\mathbf{H}_1^f(\mathbf{X}_i)$ is fully connected to the corresponding $n_1$-dimensional vector in $\mathbf{H}_1^f(\mathbf{X}_{i-1})$ at the previous time step. Therefore, the patch connections of BRCN are actually those of a "discriminative" TRBM. In other words, by setting the filter sizes of feedforward and conditional convolutions as the size of the whole frame, BRCN is equivalent to TRBM.

Compared with TRBM, BRCN has the following advantages for handling the task of video super-resolution. 1) BRCN restricts the receptive field of original full connection to a patch rather than the whole frame, which can capture the temporal change of visual details. 2) BRCN replaces all the full connections with weight-sharing convolutional ones, which largely reduces the computational cost. 3) BRCN is more flexible to handle videos of different sizes, once it is trained on a fixed-size video dataset. Similar to TRBM, the proposed model can be generalized to other sequence modelling applications, e.g., video motion modelling [22].

### 3.3 Network Learning

Through combining Equations 1, 2 and 3, we can obtain the desired prediction $\mathbf{O}(\mathcal{X}; \Theta)$ from the low-resolution video $\mathcal{X}$, where $\Theta$ denotes the network parameters. Network learning proceeds by minimizing the Mean Square Error (MSE) between the predicted high-resolution video $\mathbf{O}(\mathcal{X}; \Theta)$ and the groundtruth $\mathcal{Y}$:

$$\mathcal{L} = \|\mathbf{O}(\mathcal{X}; \Theta) - \mathcal{Y}\|^2 \tag{4}$$

via stochastic gradient descent. Actually, stochastic gradient descent is enough to achieve satisfying results, although we could exploit other optimization algorithms with more computational cost, e.g., L-BFGS. During optimization, all the filter weights of recurrent and conditional convolutions are initialized by randomly sampling from a Gaussian distribution with mean 0 and standard deviation 0.001, whereas the filter weights of feedforward convolution are pre-trained on static images [6]. Note that the pretraining step only aims to speed up training by providing a better parameter initialization, due to the limited size of training set. This step can be avoided by alternatively using a larger scale dataset. We experimentally find that using a smaller learning rate (e.g., $1e-4$) for the weights in the output layer is crucial to obtain good performance.

## 4 Experimental Results

To verify the effectiveness, we apply the proposed model to the task of video SR, and present both quantitative and qualitative results as follows.

### 4.1 Datasets and Implementation Details

We use 25 YUV format video sequences[2] as our training set, which have been widely used in many video SR methods [13, 16, 21]. To enlarge the training set, model training is performed in a volume-based way, i.e., cropping multiple overlapped volumes from training videos and then regarding each volume as a training sample. During cropping, each volume has a spatial size of $32 \times 32$ and a temporal step of 10. The spatial and temporal strides are 14 and 8, respectively. As a result, we can generate roughly 41,000 volumes from the original dataset. We test our model on a variety of challenging videos, including *Dancing*, *Flag*, *Fan*, *Treadmill* and *Turbine* [19], which contain complex motions with severe motion blur and aliasing. Note that we do not have to extract volumes during testing, since the convolutional operation can scale to videos of any spatial size and temporal step. We generate the testing dataset with the following steps: 1) using Gaussian filter with standard deviation 2 to smooth each original frame, and 2) downsampling the frame by a factor of 4 with bicubic method[3].

Table 1: The results of PSNR (dB) and running time (sec) on the testing video sequences.

| Video | Bicubic | | SC [25] | | K-SVD [26] | | NE+NNLS [4] | | ANR [23] | |
|---|---|---|---|---|---|---|---|---|---|---|
| | PSNR | Time | PSNR | Time | PSNR | Time | PSNR | Time | PSNR | Time |
| *Dancing* | 26.83 | - | 26.80 | 45.47 | 27.69 | 2.35 | 27.63 | 19.89 | 27.67 | 0.85 |
| *Flag* | 26.35 | - | 26.28 | 12.89 | 27.61 | 0.58 | 27.41 | 4.54 | 27.52 | 0.20 |
| *Fan* | 31.94 | - | 32.50 | 12.92 | 33.55 | 1.06 | 33.45 | 8.27 | 33.49 | 0.38 |
| *Treadmill* | 21.15 | - | 21.27 | 15.47 | 22.22 | 0.35 | 22.08 | 2.60 | 22.24 | 0.12 |
| *Turbine* | 25.09 | - | 25.77 | 16.49 | 27.00 | 0.51 | 26.88 | 3.67 | 27.04 | 0.18 |
| Average | 26.27 | - | 26.52 | 20.64 | 27.61 | 0.97 | 27.49 | 7.79 | 27.59 | 0.35 |

| Video | NE+LLE [5] | | SR-CNN [6] | | 3DSKR [21] | | Enhancer [1] | | BRCN | |
|---|---|---|---|---|---|---|---|---|---|---|
| | PSNR | Time | PSNR | Time | PSNR | Time | PSNR | Time | PSNR | Time |
| *Dancing* | 27.64 | 4.20 | 27.81 | 1.41 | 27.81 | 1211 | 27.06 | - | **28.09** | 3.44 |
| *Flag* | 27.48 | 0.96 | 28.04 | 0.36 | 26.89 | 255 | 26.58 | - | **28.55** | 0.78 |
| *Fan* | 33.46 | 1.76 | 33.61 | 0.60 | 31.91 | 323 | 32.14 | - | **33.73** | 1.46 |
| *Treadmill* | 22.22 | 0.57 | 22.42 | 0.15 | 22.32 | 127 | 21.20 | - | **22.63** | 0.46 |
| *Turbine* | 26.98 | 0.80 | 27.50 | 0.23 | 24.27 | 173 | 25.60 | - | **27.71** | 0.70 |
| Average | 27.52 | 1.66 | 27.87 | 0.55 | 26.64 | 418 | 26.52 | - | **28.15** | 1.36 |

Table 2: The results of PSNR (dB) by variants of BRCN on the testing video sequences. $v$: feedforward convolution, $r$: recurrent convolution, $t$: conditional convolution, $b$: bidirectional scheme.

| Video | BRCN $\{v\}$ | BRCN $\{v,r\}$ | BRCN $\{v,t\}$ | BRCN $\{v,r,t\}$ | BRCN $\{v,r,t,b\}$ |
|---|---|---|---|---|---|
| *Dancing* | 27.81 | 27.98 | 27.99 | 28.09 | 28.09 |
| *Flag* | 28.04 | 28.32 | 28.39 | 28.47 | 28.55 |
| *Fan* | 33.61 | 33.63 | 33.65 | 33.65 | 33.73 |
| *Treadmill* | 22.42 | 22.59 | 22.56 | 22.59 | 22.63 |
| *Turbine* | 27.50 | 27.47 | 27.50 | 27.62 | 27.71 |
| Average | 27.87 | 27.99 | 28.02 | 28.09 | 28.15 |

Some important parameters of our network are illustrated as follows: $f_{v_1}$=9, $f_{v_3}$=5, $n_1$=64, $n_2$=32 and $c$=1[4]. Note that varying the number and size of filters does not have a significant impact on the performance, because some filters with certain sizes are already in a regime where they can almost reconstruct the high-resolution videos [24, 6].

## 4.2 Quantitative and Qualitative Comparison

We compare our BRCN with two multi-frame SR methods including 3DSKR [21] and a commercial software namely Enhancer [1], and seven single-image SR methods including Bicubic, SC [25], K-SVD [26], NE+NNLS [4], ANR [23], NE+LLE [5] and SR-CNN [6].

The results of all the methods are compared in Table 1, where evaluation measures include both peak signal-to-noise ratio (PSNR) and running time (Time). Specifically, compared with the state-of-the-art single-image SR methods (e.g., SR-CNN, ANR and K-SVD), our multi-frame-based method can surpass them by 0.28∼0.54 dB, which is mainly attributed to the beneficial mechanism of temporal dependency modelling. BRCN also performs much better than the two representative multi-frame SR methods (3DSKR and Enhancer) by 1.51 dB and 1.63 dB, respectively. In fact, most existing multi-frame-based methods tend to fail catastrophically when dealing with very complex motions, because it is difficult for them to estimate the motions with pinpoint accuracy.

For the proposed BRCN, we also investigate the impact of model architecture on the performance. We take a simplified network containing only feedforward convolution as a benchmark, and then study its several variants by successively adding other operations including bidirectional scheme, recurrent and conditional convolutions. The results by all the variants of BRCN are shown in Table 2, where the elements in the brace represent the included operations. As we can see, due to the ben-

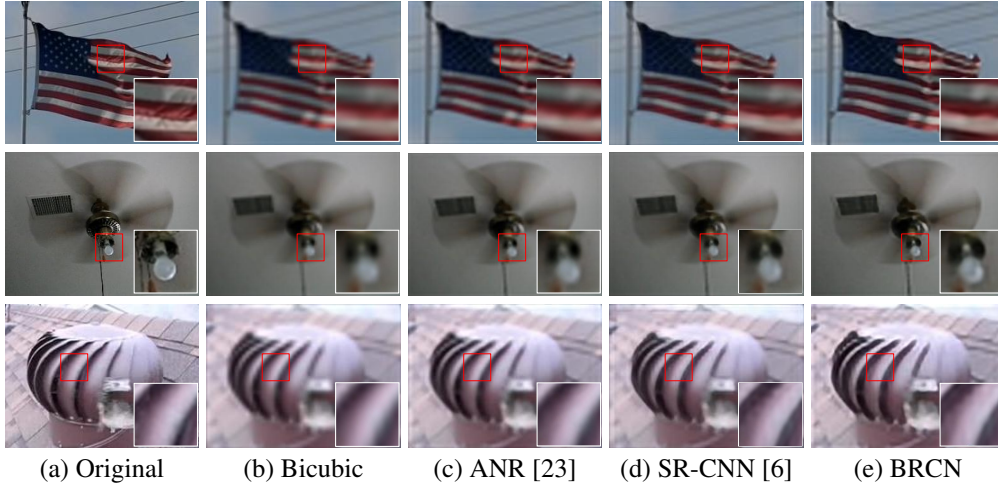

| (a) Original | (b) Bicubic | (c) ANR [23] | (d) SR-CNN [6] | (e) BRCN |

Figure 3: Closeup comparison among original frames and super resolved results by Bicubic, ANR, SR-CNN and BRCN, respectively.

efit of learning temporal dependency, exploiting either recurrent convolution $\{v, r\}$ or conditional convolution $\{v, t\}$ can greatly improve the performance. When combining these two convolutions together $\{v, r, t\}$, they obtain much better results. The performance can still be further promoted when adding the bidirectional scheme $\{v, r, t, b\}$, which results from the fact that each video frame is related to not only its previous frame but also the future one.

In addition to the quantitative evaluation, we also present some qualitative results in terms of single-frame (in Figure 3) and multi-frame (in Figure 5). **Please enlarge and view these figures on the screen for better comparison**. From these figures, we can observe that our method is able to recover more image details than others under various motion conditions.

## 4.3 Running Time

We present the comparison of running time in both Table 1 and Figure 4, where all the methods are implemented on the same machine (Intel CPU 3.10 GHz and 32 GB memory). The publicly available codes of compared methods are all in MATLAB while SR-CNN and ours are in Python. From the table and figure, we can see that our BRCN takes 1.36 sec per frame on average, which is orders of magnitude faster than the fast multi-frame SR method 3DSKR. It should be noted that the speed gap is not caused by the different MAT-LAB/Python implementations. As stated in [13, 21], the computational bottleneck for existing multi-frame SR methods is the accurate motion estimation, while our model explores an alternative based on efficient spatial-temporal convolutions which has lower computational complexity. Note that the speed of our method is worse than the fastest single-image SR method ANR. It is likely that our method involves the additional phase of temporal dependency modelling but we achieve better performance (28.15 vs. 27.59 dB).

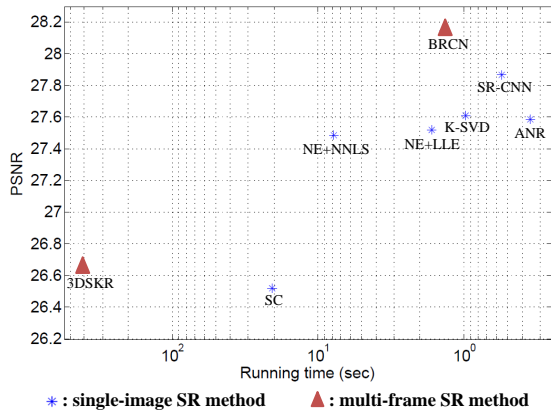

Figure 4: Running time vs. PSNR for all the methods.

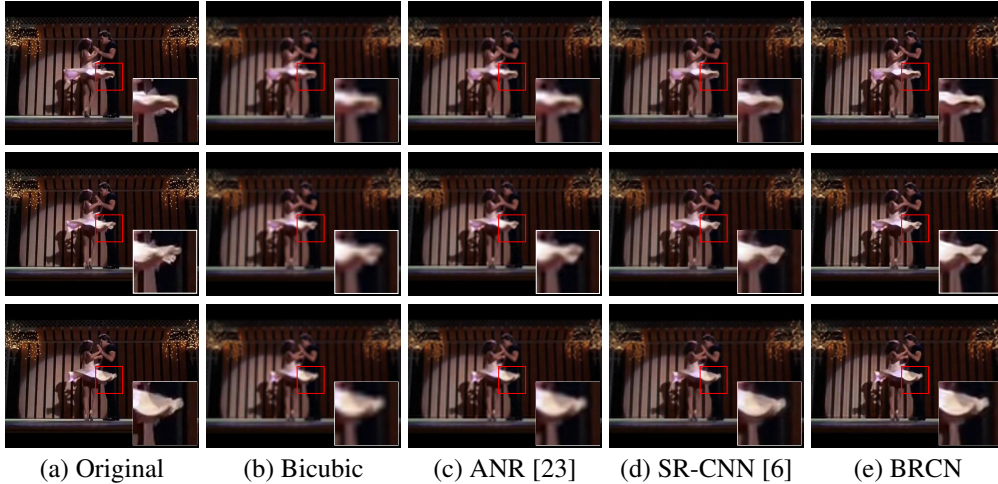

| (a) Original | (b) Bicubic | (c) ANR [23] | (d) SR-CNN [6] | (e) BRCN |

Figure 5: Comparison among original frames ($2^{th}$, $3^{th}$ and $4^{th}$ frames, from the top row to the bottom) of the *Dancing* video and super resolved results by Bicubic, ANR, SR-CNN and BRCN, respectively.

## 4.4 Filter Visualization

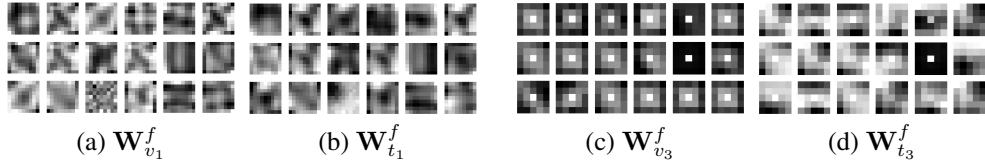

| (a) $\mathbf{W}^f_{v_1}$ | (b) $\mathbf{W}^f_{t_1}$ | (c) $\mathbf{W}^f_{v_3}$ | (d) $\mathbf{W}^f_{t_3}$ |

Figure 6: Visualization of learned filters by the proposed BRCN.

We visualize the learned filters of feedforward and conditional convolutions in Figure 6. The filters of $\mathbf{W}^f_{v_1}$ and $\mathbf{W}^f_{t_1}$ exhibit some strip-like patterns, which can be viewed as edge detectors. The filters of $\mathbf{W}^f_{v_3}$ and $\mathbf{W}^f_{t_3}$ show some centrally-averaging patterns, which indicate that the predicted high-resolution frame is obtained by averaging over the feature maps in the second hidden layer. This averaging operation is also in consistent with the corresponding reconstruction phase in patch-based SR methods (e.g., [25]), but the difference is that our filters are automatically learned rather than pre-defined. When comparing the learned filters between feedforward and conditional convolutions, we can also observe that the patterns in the filters of feedforward convolution are much more regular and clear.

## 5 Conclusion and Future Work

In this paper, we have proposed the bidirectional recurrent convolutional network (BRCN) for multi-frame SR. Our main contribution is the novel use of bidirectional scheme, recurrent and conditional convolutions for temporal dependency modelling. We have applied our model to super resolve videos containing complex motions, and achieved better performance and faster speed. In the future, we will perform comparisons with other multi-frame SR methods.

### Acknowledgments

This work is jointly supported by National Natural Science Foundation of China (61420106015, 61175003, 61202328, 61572504) and National Basic Research Program of China (2012CB316300).

## Footnotes

[1]Note that we upscale each low-resolution frame in the sequence to the desired size with bicubic interpolation in advance.

[2]http://www.codersvoice.com/a/webbase/video/08/152014/130.html.

[3]Here we focus on the factor of 4, which is usually considered as the most difficult case in super-resolution.

[4]Similar to [23], we only deal with luminance channel in the YCrCb color space. Note that our model can be generalized to handle all the three channels by setting $c$=3. Here we simply upscale the other two channels with bicubic method for well illustration.

# References

[1] Video enhancer. http://www.infognition.com/videoenhancer/, version 1.9.10. 2014.

[2] S. Baker and T. Kanade. Super-resolution optical flow. *Technical report, CMU*, 1999.

[3] B. Bascle, A. Blake, and A. Zisserman. Motion deblurring and super-resolution from an image sequence. *European Conference on Computer Vision*, pages 571–582, 1996.

[4] M. Bevilacqua, A. Roumy, C. Guillemot, and M.-L. A. Morel. Low-complexity single-image super-resolution based on nonnegative neighbor embedding. *British Machine Vision Conference*, 2012.

[5] H. Chang, D.-Y. Yeung, and Y. Xiong. Super-resolution through neighbor embedding. *IEEE Conference on Computer Vision and Pattern Recognition*, page I, 2004.

[6] C. Dong, C. C. Loy, K. He, and X. Tang. Learning a deep convolutional network for image super-resolution. *European Conference on Computer Vision*, pages 184–199, 2014.

[7] D. Eigen, D. Krishnan, and R. Fergus. Restoring an image taken through a window covered with dirt or rain. *IEEE International Conference on Computer Vision*, pages 633–640, 2013.

[8] W. T. Freeman, E. C. Pasztor, and O. T. Carmichael. Learning low-level vision. *International Journal of Computer Vision*, pages 25–47, 2000.

[9] D. Glasner, S. Bagon, and M. Irani. Super-resolution from a single image. *IEEE International Conference on Computer Vision*, pages 349–356, 2009.

[10] M. Irani and S. Peleg. Improving resolution by image registration. *CVGIP: Graphical Models and Image Processing*, pages 231–239, 1991.

[11] V. Jain and S. Seung. Natural image denoising with convolutional networks. *Advances in Neural Information Processing Systems*, pages 769–776, 2008.

[12] K. Jia, X. Wang, and X. Tang. Image transformation based on learning dictionaries across image spaces. *IEEE Transactions on Pattern Analysis and Machine Intelligence*, pages 367–380, 2013.

[13] C. Liu and D. Sun. On bayesian adaptive video super resolution. *IEEE Transactions on Pattern Analysis and Machine Intelligence*, pages 346–360, 2014.

[14] D. Mitzel, T. Pock, T. Schoenemann, and D. Cremers. Video super resolution using duality based tv-l 1 optical flow. *Pattern Recognition*, pages 432–441, 2009.

[15] V. Nair and G. E. Hinton. Rectified linear units improve restricted boltzmann machines. *International Conference on Machine Learning*, pages 807–814, 2010.

[16] M. Protter, M. Elad, H. Takeda, and P. Milanfar. Generalizing the nonlocal-means to super-resolution reconstruction. *IEEE Transactions on Image Processing*, pages 36–51, 2009.

[17] R. R. Schultz and R. L. Stevenson. Extraction of high-resolution frames from video sequences. *IEEE Transactions on Image Processing*, pages 996–1011, 1996.

[18] M. Schusterand and K. K. Paliwal. Bidirectional recurrent neural networks. *IEEE Transactions on Signal Processing*, pages 2673–2681, 1997.

[19] O. Shahar, A. Faktor, and M. Irani. Space-time super-resolution from a single video. *IEEE Conference on Computer Vision and Pattern Recognition*, pages 3353–3360, 2011.

[20] I. Sutskever and G. E. Hinton. Learning multilevel distributed representations for high-dimensional sequences. In *International Conference on Artificial Intelligence and Statistics*, pages 548–555, 2007.

[21] H. Takeda, P. Milanfar, M. Protter, and M. Elad. Super-resolution without explicit subpixel motion estimation. *IEEE Transactions on Image Processing*, pages 1958–1975, 2009.

[22] G. Taylor, G. Hinton, and S. Roweis. Modeling human motion using binary latent variables. *Advances in Neural Information Processing Systems*, pages 448–455, 2006.

[23] R. Timofte, V. De, and L. V. Gool. Anchored neighborhood regression for fast example-based super-resolution. *IEEE International Conference on Computer Vision*, pages 1920–1927, 2013.

[24] L. Xu, J. S. Ren, C. Liu, and J. Jia. Deep convolutional neural network for image deconvolution. In *Advances in Neural Information Processing Systems*, pages 1790–1798, 2014.

[25] J. Yang, J. Wright, T. S. Huang, and Y. Ma. Image super-resolution via sparse representation. *IEEE Transactions on Image Processing*, pages 2861–2873, 2010.

[26] R. Zeyde, M. Elad, and M. Protte. On single image scale-up using sparse-representations. *Curves and Surfaces*, pages 711–730, 2012.

